# Neural Network Methods for Optimization Problems

**Arun Jagota**
Department of Mathematical Sciences
Memphis State University
Memphis, TN 38152
E-mail: jagota@next1.msci.memst.edu

In a talk entitled "Trajectory Control of Convergent Networks with applications to TSP", Natan Peterfreund (Computer Science, Technion) dealt with the problem of controlling the trajectories of continuous convergent neural networks models for solving optimization problems, without affecting their equilibria set and their convergence properties. Natan presented a class of feedback control functions which achieve this objective, while also improving the convergence rates. A modified Hopfield and Tank neural network model, developed through the proposed feedback approach, was found to substantially improve the results of the original model in solving the Traveling Salesman Problem. The proposed feedback overcame the 2n symmetric property of the TSP problem.

In a talk entitled "Training Feedforward Neural Networks quickly and accurately using Very Fast Simulated Reannealing Methods", Bruce Rosen (Asst. Professor, Computer Science, UT San Antonio) presented the Very Fast Simulated Reannealing (VFSR) algorithm for training feedforward neural networks [2]. VFSR Trained networks avoid getting stuck in local minima and statistically guarantee the finding of an optimal weights set. The method can be used when network activation functions are nondifferentiable, and although often slower than gradient descent, it is faster than other Simulated Annealing methods. The performances of conjugate gradient descent and VFSR trained networks were demonstrated on a set of difficult logic problems.

In a talk entitled "A General Method for Finding Solutions of Covering problems by Neural Computation", Tal Grossman (Complex Systems, Los Alamos) presented a neural network algorithm for finding small minimal covers of hypergraphs. The network has two sets of units, the first representing the hyperedges to be covered and the second representing the vertices. The connections between the units are determined by the edges of the incidence graph. The dynamics of these two types of units are different. When the parameters of the units are correctly tuned, the stable states of the system correspond to the possible covers. As an example, he found new large square free subgraphs of the hypercube.

In a talk entitled "Algebraic and Grammatical Design of Relaxation Nets", Eric

Mjolsness (Professor, Computer Science, Yale University) presented useful algebraic notation and computer-algebraic syntax for general "programming" with optimization ideas; and also some optimization methods that can be succinctly stated in the proposed notation. He addressed global versus local optimization, time and space cost, learning, expressiveness and scope, and validation on applications. He discussed the methods of algebraic expression (optimization syntax and transformations, grammar models), quantitative methods (statistics and statistical mechanics, multiscale algorithms, optimization methods), and the systematic design approach.

In a talk entitled "Algorithms for Touring Knights", Ian Parberry (Associate Professor, Computer Sciences, University of North Texas) compared Takefuji and Lee's neural network for knight's tours with a random walk and a divide-and-conquer algorithm. The experimental and theoretical evidence indicated that the neural network is the slowest approach, both on a sequential computer and in parallel, and for the problems of generating a single tour, and generating as many tours as possible.

In a talk entitled "Report on the DIMACS Combinatorial Optimization Challenge", Arun Jagota (Asst. Professor, Math Sciences, Memphis State University) presented his work, towards the said challenge, on neural network methods for the fast approximate solution of the Maximum Clique problem. The *Mean Field Annealing* algorithm was implemented on the Connection Machine CM-5. A fast (two-temperature) annealing schedule was experimentally evaluated on random graphs and on the challenge benchmark graphs, and was shown to work well. Several other algorithms, of the randomized local search kind, including one employing *reinforcement learning* ideas, were also evaluated on the same graphs. It was concluded that the neural network algorithms were in the middle in the solution quality versus running time trade-off, in comparison with a variety of conventional methods.

In a talk entitled "Optimality in Biological and Artificial Networks", Daniel Levine (Professor, Mathematics, UT Arlington) previewed a book to appear in 1995 [1]. Then he expanded his own view, that human cognitive functioning is sometimes, but not always or even most of the time, optimal. There is a continuum from the most "disintegrated" behavior, associated with frontal lobe damage, to stereotyped or obsessive-compulsive behavior, to entrenched neurotic and bureaucratic habits, to rational maximization of some measurable criteria, and finally to the most "integrated", self-actualization (Abraham Maslow's term) which includes both reason and intuition. He outlined an alternative to simulated annealing, whereby a network that has reached an energy minimum in some but not all of its variables can move out of it through a "negative affect" signal that responds to a comparison of energy functions between the current state and imagined alternative states.

## References

[1] D.S. Levine & W. Elsberry, editors. *Optimality in Biological and Artificial Networks?* Lawrence Erlbaum Associates, 1995.

[2] B. E. Rosen & J. M. Goodwin. Training hard to learn networks using advanced simulated annealing methods. In *Proc. of ACM Symp. on Applied Comp.*.
